# Learning Theory and Experiments with Competitive Networks

**Griff L. Bilbro**
North Carolina State University
Box 7914
Raleigh, NC 27695-7914

**David E. Van den Bout**
North Carolina State University
Box 7914
Raleigh, NC 27695-7914

## Abstract

We apply the theory of Tishby, Levin, and Solla (TLS) to two problems. First we analyze an elementary problem for which we find the predictions consistent with conventional statistical results. Second we numerically examine the more realistic problem of training a competitive net to learn a probability density from samples. We find TLS useful for predicting average training behavior.

## 1 TLS APPLIED TO LEARNING DENSITIES

Recently a theory of learning has been constructed which describes the learning of a relation from examples (Tishby, Levin, and Solla, 1989), (Schwartz, Samalan, Solla, and Denker, 1990). The original derivation relies on a statistical mechanics treatment of the probability of independent events in a system with a specified average value of an additive error function.

The resulting theory is not restricted to learning relations and it is not essentially statistical mechanical. The TLS theory can be derived from the *principle of maximum entropy*, a general inference tool which produces probabilities characterized by certain values of the averages of specified functions(Jaynes, 1979). A TLS theory can be constructed whenever the specified function is additive and associated with independent examples. In this paper we treat the problem of learning a probability density from samples.

Consider the model as some function $p(x|\omega)$ of fixed form and adjustable parameters $\omega$ which are to be chosen to approximate $\overline{p}(x)$ where the overline denotes the true density. All we know about $\overline{p}$ are the elements of a training set $T$ which are drawn from it. Define an error $\epsilon(x|\omega)$. By the principal of maximum entropy

$$p(x|\omega) = \frac{1}{z(\beta)} e^{-\beta\epsilon(x|\omega)}, \qquad (1)$$

can be interpreted as the unique density which contains no other information except a specified value of the average error

$$\langle\epsilon\rangle = \int dx \; p(x|\omega)\epsilon(x|\omega). \qquad (2)$$

In Equation 1 $z$ is a normalization that is assumed to be independent of the value of $\omega$; the parameter $\beta$ is called the *sensitivity* and is adjusted so that the average error is equal to some $\epsilon_T$, the specified target error on the training set. We will use the convention that an integral operates on the entire expression that follows it.

The usual Bayes rule produces a density in $\omega$ from $p(x|\omega)$ and from a prior density $\rho^{(0)}(\omega)$ which reflects at best a genuine prior probability or at least a restriction to the acceptable portion of the search space. Posterior to training on $m$ certain examples,

$$\rho^{(m)}(\omega) = \frac{\rho^{(0)}(\omega)}{Z_m} \exp(-\beta \sum_{i=1}^{m} \epsilon(x_i|\omega)) \qquad (3)$$

where $Z_m$ is a normalization that depends on the particular set of examples as well as their number. In order to remove the effect of any particular set of examples, we can average this posterior density over all possible $m$ examples

$$\langle\rho^{(m)}(\omega)\rangle_x = \int dx^{(m)} \overline{p}(x_1)\overline{p}(x_2)...\overline{p}(x_m)\rho^{(m)}(\omega). \qquad (4)$$

This average posterior density models the expected density of nets or $\omega$ after training. This distribtution in $\omega$ implies the following expected posterior density for a new example $x_{m+1}$

$$\int d\omega p(x_{m+1}|\omega)\langle\rho^{(m)}(\omega)\rangle_x. \qquad (5)$$

TLS compare this probability in $x_{m+1}$ with the true target probability to obtain the *Average Prediction Probability* or APP after training

$$p^{(m)} = \int dx_{m+1}\overline{p}(x_{m+1}) \int d\omega p(x_{m+1}|\omega)\langle\rho^{(m)}(\omega)\rangle_x, \qquad (6)$$

the average over both the training set $x^{(m)}$ and an independent test example $x_{m+1}$.

In the averages of Equations 4 and 6 are inconvenient to evaluate exactly because of the $Z_m$ term in Equation 3. TLS propose an "annealed approximation" to APP in which the average of the ratio of Equation 4 is replaced by the ratio of the averages. Equation 6 becomes

$$p^{(m)} = \frac{\int d\omega \rho^{(0)}(\omega)g^{m+1}(\omega)}{\int d\omega \rho^{(0)}(\omega)g^m(\omega)} \qquad (7)$$

where

$$g(\omega) = \int dx \overline{p}(x)p(x|\omega). \qquad (8)$$

Equation 7 is well suited for theoretical analysis and is also convenient for numerical predictions. To apply Equation 7 numerically, we will produce Monte Carlo estimates for the moments of $g$ that involve sampling $\rho^{(0)}(\omega)$. If the dimension of $\omega$ is larger than 50, it is preferable to histogram $g$ rather than evaluate the moments directly.

## 1.1   ANALYSIS OF AN ELEMENTARY EXAMPLE

In this section we theoretically analyze a learning problem with the TLS theory. We will study the adjustment of the mean of a Gaussian density to represent a finite number of samples. The utility of this elementary example is that it admits an analytic solution for the APP of the previous section. All the relevant integrals can be computed with the identity

$$\int_{-\infty}^{\infty} dx \, \exp\left(-a_1(x-b_1)^2 - a_2(x-b_2)^2\right) = \sqrt{\frac{\pi}{a_1+a_2}} \, \exp\left(-\frac{a_1 a_2}{a_1+a_2}(b_1-b_2)^2\right).$$

(9)

We take the true density to be a Gaussian of mean $\overline{\omega}$ and variance $1/2\alpha$

$$\overline{p}(x) = \sqrt{\frac{\alpha}{\pi}} e^{-\alpha(x-\overline{\omega})^2}.$$

(10)

We model the prior density as a Gaussian with mean $\omega_0$ and variance $1/2r$

$$\rho^{(0)}(\omega) = \sqrt{\frac{r}{\pi}} e^{-r(\omega-\omega_0)^2}.$$

(11)

We choose the simplest error function

$$\epsilon(x|\omega) = (x-\omega)^2,$$

(12)

the squared error between a sample $x$ and the Gaussian "model" defined by its mean $\omega$, which is to become our estimate of $\overline{\omega}$. In Equation 1, this error function leads to

$$p(x|\omega) = \frac{1}{z(\beta)} e^{-\beta(x-\omega)^2}$$

(13)

with $z(\beta) = \sqrt{\frac{\pi}{\beta}}$ which is independent of $\omega$ as assumed. We determine $\beta$ by solving for the error on the training set to get $\beta = \frac{1}{2\epsilon_T}$.

The generalization, Equation 8, can now be evaluated with Equation 9

$$g(\omega) = \sqrt{\frac{\kappa}{\pi}} e^{-\kappa(\omega-\overline{\omega})^2},$$

(14)

where

$$\kappa = \frac{\alpha\beta}{\alpha+\beta},$$

(15)

is less than either $\alpha$ or $\beta$. The denominator of Equation 7 becomes

$$\left(\frac{\kappa}{\pi}\right)^{m/2} \sqrt{\frac{r}{r+m\kappa}} \, \exp\left(-\frac{m\kappa r}{m\kappa+r}(\overline{\omega}-\omega_0)^2\right)$$

(16)

with a similar expression for the numerator.

The case of many examples or little prior knowledge is interesting. Consider Equations 7 and 16 in the limit $m\kappa \gg r$

$$p^{(m)} = \sqrt{\frac{\kappa}{\pi}}\sqrt{\frac{m}{m+1}}, \qquad (17)$$

which climbs to an asymptotic value of $\sqrt{\frac{\kappa}{\pi}}$ for $m \longrightarrow \infty$. In order to compare this with intuition, consider that the sample mean of $\{x_1, x_2, ..., x_m\}$ approaches $\bar{w}$ to within a variance of $1/2m\alpha$, so that

$$\langle \rho^{(m)}(\omega) \rangle_x \approx \sqrt{\frac{m\alpha}{\pi}} e^{-m\alpha(x-\bar{w})^2} \qquad (18)$$

which makes Equation 6 agree with Equation 17 for large enough $\beta$. In this sense, the statistical mechanical theory of learning differs from conventional Bayesian estimation only in its choice of an unconventional performance criterion APP.

## 2   GENERAL NUMERICAL PROCEDURE

In this section we apply the theory to the more realistic problem of learning a continuous probability density from a finite sample set. We can estimate the moments of Equation 7 by the following Monte Carlo procedure. Given a training set $\mathcal{T} = \{x_t\}_{t=1}^{t=T}$ drawn from the unknown density $\bar{p}$ on domain $X$ with finite volume $V$, an error function $\epsilon(x|\omega)$, a training error $\epsilon_T$, and a prior density $\rho^{(0)}(\omega)$ of vectors such that each $\omega$ specifies a candidate function,

1. Construct two sample sets: a prior set of $P$ functions $\mathcal{P} = \{\omega_p\}$ drawn from $\rho^{(0)}(\omega)$ and a set of $U$ input vectors $\mathcal{U} = \{x_u\}$ drawn uniformly from $X$. For each $p$ in the prior set, tabulate the error $\epsilon_{up} = \epsilon(x_u|\omega_p)$ for every point in $\mathcal{U}$ and the error $\epsilon_{tp} = \epsilon(x_t|\omega_p)$ for every point in $\mathcal{T}$.

2. Determine the sensitivity $\beta$ by solving the equation $\langle \epsilon \rangle = \epsilon_T$ where

$$\langle \epsilon \rangle = \frac{\sum_u e^{-\beta \epsilon_{up}} \epsilon_{up}}{\sum_u e^{-\beta \epsilon_{up}}}. \qquad (19)$$

3. Estimate the average generalization of a given $\omega_p$ from Equation 8

$$g(\omega_p) = \frac{1}{V} \frac{1/T \sum_t e^{-\beta \epsilon_{tp}}}{1/U \sum_u e^{-\beta \epsilon_{up}}}. \qquad (20)$$

4. The performance after $m$ examples is the ratio of Equation 7. By construction $\mathcal{P}$ is drawn from $\rho^{(0)}$ so that

$$p^{(m)} = \frac{\sum_p g^{m+1}(\omega_p)}{\sum_p g^m(\omega_p)}. \qquad (21)$$

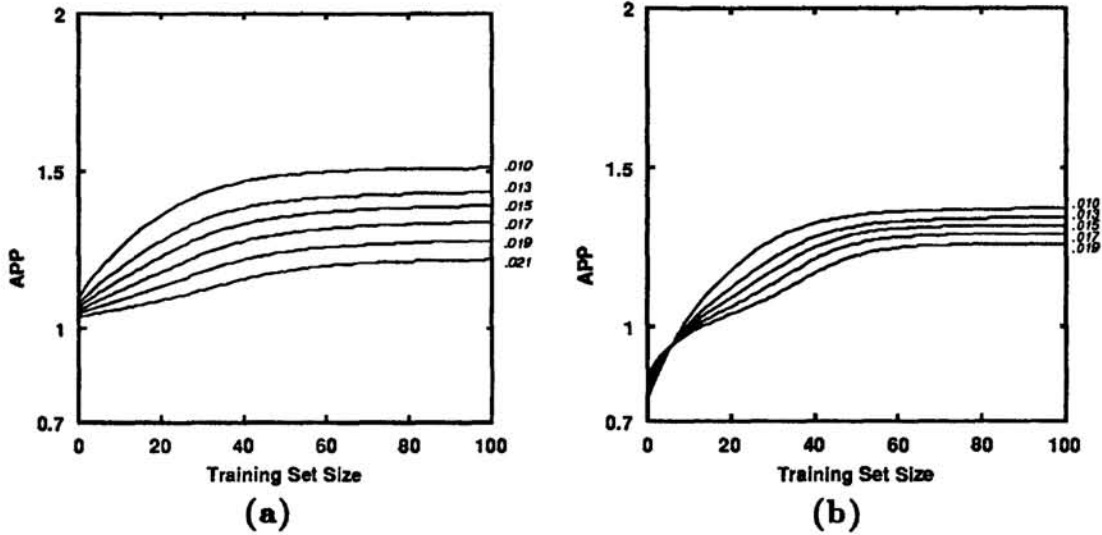

Figure 1: Predicted APP versus number of training samples for a 20-neuron competitive network trained to various target errors where the neuron weights were initialized from (a) a uniform density, (b) an antisymmetrically skewed density.

## 3   COMPETITIVE LEARNING NETS

We consider competitive learning nets (CLNs) because they are familiar and useful to us (Van den Bout and Miller, 1990), because there exist two widely known training strategies for CLNs (the neurons can learn either independently or under a global interaction called conscience (DeSieno, 1988), and because CLNs can be applied to one-dimensional problems without being too trivial. Competitive learning nets with conscience qualitatively change their behavior when they are trained on finite sample sets containing fewer examples than neurons; except for that regime we found the theory satisfactory. All experiments in this section were conducted upon the following one-dimensional training density

$$\bar{p}(x) = \begin{cases} \frac{1}{2\sqrt{x}} & 0 \le x \le 1, \\ 0 & \text{otherwise.} \end{cases}$$

In Figure 1 is the Average Prediction Probability (APP) for $k = 20$ versus $m$, for several values of target error $\epsilon_T$ and for two prior densitsities; first consider predictions from the uniform prior. For $\epsilon_T = 0.01$, APP practically attains its asymptote of 1.5 by $m = 40$ examples. Assuming the APP to be dominated in the limit by the largest $g$, we expect a CLN trained to an error of 0.01 on a set of 40 examples to perform 1.5 times better than an untrained net on unseen samples drawn from the same probability density. This leads to a predicted probable error of about

$$\epsilon_{prob} = \frac{1}{2\,k\,p^{(m)}}. \tag{22}$$

For $k = 20$, $\epsilon_{prob} = 0.017$ for $\epsilon_T = .01$ and $\epsilon_{prob} = 0.021$ for $\epsilon_T = 0.02$.

We performed 5,000 training trials of a 20-neuron CLN on randomly selected sets of

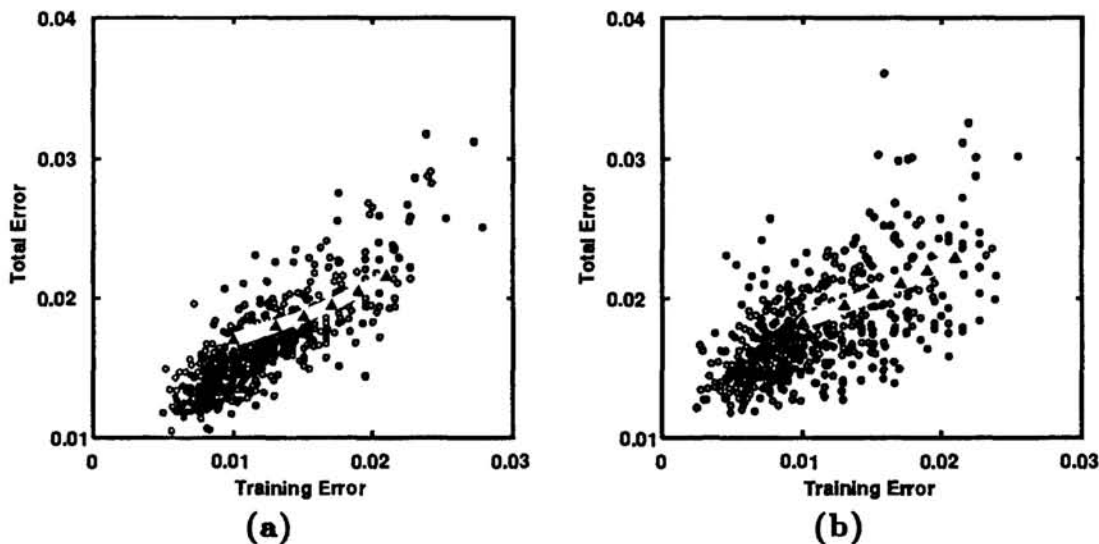

Figure 2: Experimentally determined and predicted values of total error across the training density after competitive learning was performed using a 20-neuron network trained to various target errors (a) with 40 samples, (b) with 20 samples.

40 samples from the training density. Each network was trained to a target error in the range $[0.005, 0.03]$ on its 40 samples, and the average error on the total density was then calculated for the trained network. Figure 2 is a plot of 500 of these trials along with the predicted errors for various target errors. The probable error is qualitatively correct and the scatter of actual experiments increases in width by about the ratio of APPs for $m = 20$ and $m = 40$. For the case of $m = 20$ examples, the same net can only be expected to exhibit probable errors of .019 and .023 for corresponding training target errors, which is compared graphically in Figure 2 with the experimentally determined errors for $m = 20$.

The APP curves saturate at a value of $m$ that is insensitive to the prior density from which the nets are drawn. The vertical scale does depend somewhat on the prior however. Consider Figure 1, which also shows the APP curves for the same $k = 20$ net with the prior density antisymmetrically skewed *away* from the true density by the following function:

$$\rho^{(0)}(\omega) = \left\{ \begin{array}{ll} \frac{1}{2\sqrt{1-\omega}} & 0 \leq \omega \leq 1, \\ 0 & \text{otherwise.} \end{array} \right.$$

For $m > 20$ the *shapes* of the curves are almost unchanged, even though the vertical scale is different: saturation occurs at about the same value of $m$. Even when the prior greatly overrepresents poor nets, their effect on the prediction rapidly diminishes with training set size. This is important because in actual training, the effect of the initial configuration is also quickly lost. For $m < 20$ the predictions are not valid in any case, since our simple error function does not reflect the actual probability even approximately for $m < k$ in these nets. It is for $m < 20$ where the only significant differences between the two families of curves occur. We have also been able to draw the same conclusions from less structured prior densities generated by assigning positive normalized random numbers to intervals of the

domain. Moreover, we generally find that TLS predicts that about twice as many samples as neurons are needed to train competitive nets of other sizes.

# 4   CONCLUSION

TLS can be applied to learning densities as well as relations. We considered the effects of varying the number of examples, the target training error, and the choice of prior density. In these experiments on learning a density as well as others dealing with learning a binary output (Bilbro and Snyder, 1990), a ternary output (Chow, Bilbro, and Yee, 1990), and a continuous output (Bilbro and Klenin, 1990) we find if saturation occurs for $m$ substantially less than the total number of available samples, say $m < |T|/2$, that $m$ is a good predictor of sufficient training set size. Moreover there is evidence from a reformulation of the learning theory based on the grand canonical ensemble that supports this statistical approach (Klenin,1990).

## References

G. L. Bilbro and M. Klenin. (1990) Thermodynamic Models of Learning: Applications. Unpublished.

G. L. Bilbro and W. E. Snyder. (1990) Learning theory, linear separability, and noisy data. CCSP-TR-90/7, Center for Communications and Signal Processing, Box 7914, Raleigh, NC 27695-7914.

M. Y. Chow, G. L. Bilbro and S. O. Yee. (1990) Application of Learning Theory to Single-Phase Induction Motor Incipient Fault Detection Artificial Neural Networks. Submitted to *International Journal of Neural Systems.*

D. DeSieno. (1988) Adding a conscience to competitive learning. In *IEEE International Conference on Neural Networks*, pages I:117–I:124.

E. T. Jaynes. (1979) Where Do We Stand on Maximum Entropy?. In R. D. Leven and M. Tribus (Eds.), *Maximum Entropy Formalism*, M. I. T. Press, Cambridge, pages 17-118.

M. Klenin. (1990) Learning Models and Thermostatistics: A Description of Overtraining and Generalization Capacities. NETR-90/3, Center for Communications and Signal Processing, Neural Engineering Group, Box 7914, Raleigh, NC 27695-7914.

D. B. Schwartz, V. K. Samalan, S. A. Solla & J. S. Denker. (1990) Exhaustive Learning. *Neural Computation.*

N. Tishby, E. Levin, and S. A. Solla. (1989) Consistent inference of probabilities in layered networks: Predictions and generalization. *IJCNN*, IEEE, New York, pages II:403-410.

D. E. Van den Bout and T. K. Miller III. (1990) TInMANN: The integer markovian artificial neural network. Accepted for publication in the *Journal of Parallel and Distributed Computing.*
